# Clustering with the Fisher Score

**Koji Tsuda,**[*] **Motoaki Kawanabe**[†] **and Klaus-Robert Müller**[‡]
[*]AIST CBRC, 2-41-6, Aomi, Koto-ku, Tokyo, 135-0064, Japan
[†]Fraunhofer FIRST, Kekuléstr. 7, 12489 Berlin, Germany
[‡]Dept. of CS, University of Potsdam, A.-Bebel-Str. 89, 14482 Potsdam, Germany
koji.tsuda@aist.go.jp, {nabe,klaus}@first.fhg.de

## Abstract

Recently the Fisher score (or the Fisher kernel) is increasingly used as a
feature extractor for classification problems. The Fisher score is a vector
of parameter derivatives of loglikelihood of a probabilistic model. This
paper gives a theoretical analysis about how class information is pre-
served in the space of the Fisher score, which turns out that the Fisher
score consists of a few important dimensions with class information and
many nuisance dimensions. When we perform clustering with the Fisher
score, K-Means type methods are obviously inappropriate because they
make use of all dimensions. So we will develop a novel but simple clus-
tering algorithm specialized for the Fisher score, which can exploit im-
portant dimensions. This algorithm is successfully tested in experiments
with artificial data and real data (amino acid sequences).

## 1  Introduction

Clustering is widely used in exploratory analysis for various kinds of data [6]. Among
them, discrete data such as biological sequences [2] are especially challenging, because
efficient clustering algorithms e.g. K-Means [6] cannot be used directly. In such cases, one
naturally considers to map data to a vector space and perform clustering there. We call the
mapping a "feature extractor". Recently, the Fisher score has been successfully applied as a
feature extractor in supervised classification [5, 15, 14, 13, 16]. The Fisher score is derived
as follows: Let us assume that a probabilistic model $q(x|\boldsymbol{\theta})$, $x \in \mathcal{X}$ is available. Given a
parameter estimate $\hat{\boldsymbol{\theta}}$ from training samples, the Fisher score vector is obtained as

$$\boldsymbol{f}(x) = (\frac{\partial \log q(x|\hat{\boldsymbol{\theta}})}{\partial \theta_1}, \ldots, \frac{\partial \log q(x|\hat{\boldsymbol{\theta}})}{\partial \theta_d})^\top.$$

The Fisher kernel refers to the inner product in this space [5]. When combined with high
performance classifiers such as SVMs, the Fisher kernel often shows superb results [5, 14].

For successful clustering with the Fisher score, one has to investigate how original classes
are mapped into the feature space, and select a proper clustering algorithm. In this paper,
it will be claimed that the Fisher score has only a few dimensions which contains the class
information and a lot of unnecessary nuisance dimensions. So K-Means type clustering [6]
is obviously inappropriate because it takes all dimensions into account. We will propose
a clustering method specialized for the Fisher score, which exploits important dimensions
with class information. This method has an efficient EM-like alternating procedure to learn,
and has the favorable property that the resultant clusters are invariant to any invertible linear

transformation. Two experiments with an artificial data and an biological sequence data will be shown to illustrate the effectiveness of our approach.

## 2  Preservation of Cluster Structure

Before starting, let us determine several notations. Denote by $\mathcal{X}$ the domain of objects (discrete or continuous) and by $\mathcal{Y} = \{1, \ldots, c\}$ the set of class labels. The feature extraction is denoted as $\boldsymbol{f}(x) : \mathcal{X} \to \Re^d$. Let $p(x, y)$ be the underlying joint distribution and assume that the class distributions $p(x|y = k)$ are well separated, i.e. $p(y = k|x)$ is close to 0 or 1.

First of all, let us assume that the marginal distribution $p(x)$ is known. Then the problem is how to find a good feature extractor, which can preserve class information, based on the prior knowledge of $p(x)$. In the Fisher score, it amounts to finding a good parametric model $q(x|\boldsymbol{\theta}), \boldsymbol{\theta} \in \Re^d$. This problem is by no means trivial, since it is in general hard to infer anything about the possible $p(y|x)$ from the marginal $p(x)$ without additional assumptions [12].

**A loss function for feature extraction**   In order to investigate how the cluster structure is preserved, we first have to define what the class information is. We regard that the class information is completely preserved, if a set of predictors in the feature space can recover the true posterior probability $p(y|x)$. This view makes sense, because it is impossible to recover the posteriors when classes are totally mixed up. As a predictor of posterior probability in the feature space, we adopt the simplest one, i.e. a linear estimator:

$$v_k(x) = \boldsymbol{w}_k^\top \boldsymbol{f}(x) + b_k, \quad \boldsymbol{w}_k \in \Re^d, \ b_k \in \Re.$$

The prediction accuracy of $v_k(x)$ for $p(y = k|x)$ is difficult to formulate, because parameters $w_k$ and $b_k$ are learned from samples. To make the theoretical analysis possible, we consider the best possible linear predictors. Thus the loss of feature extractor $\boldsymbol{f}$ for $k$-th class is defined as

$$D_k(\boldsymbol{f}) = \min_{\boldsymbol{w} \in \Re^d, b \in \Re} E_x \left( \boldsymbol{w}^\top \boldsymbol{f}(x) + b - p(y = k|x) \right)^2, \tag{2.1}$$

where $E_x$ denote the expectation with the true marginal distribution $p(x)$. The overall loss is just the sum over all classes $D(\boldsymbol{f}) = \sum_{k=1}^{c} D_k(\boldsymbol{f})$.

Even when the full class information is preserved, i.e. $D(\boldsymbol{f}) = 0$, clustering in the feature space may not be easy, because of *nuisance dimensions* which do not contribute to clustering at all. The posterior predictors make use of an at most $c$ dimensional subspace out of the $d$-dimensional Fisher score, and the complementary subspace may not have any information about classes. K-means type methods [6] assume a cluster to be hyperspherical, which means that every dimension should contribute to cluster discrimination. For such methods, we have to try to minimize the dimensionality $d$ while keeping $D(\boldsymbol{f})$ small. When nuisance dimensions cannot be excluded, we will need a different clustering method that is robust to nuisance dimensions. This issue will be discussed in Sec. 3.

**Optimal Feature Extraction**   In the following, we will discuss how to determine $q(x|\boldsymbol{\theta})$. First, a simple but unrealistic example is shown to achieve $D(\boldsymbol{f}) = 0$, without producing nuisance dimensions at all. Let us assume that $q(x|\boldsymbol{\theta})$ is determined as a mixture model of true class distributions:

$$q_0(x|\boldsymbol{\alpha}) = \sum_{k=1}^{c-1} \alpha_k p(x|y = k) + (1 - \sum_{k=1}^{c-1} \alpha_k) p(x|y = c), \ \ \alpha \in \mathcal{A}, \tag{2.2}$$

where $\mathcal{A} = \{\boldsymbol{\alpha} : \sum_{k=1}^{c-1} \alpha_k \leq 1, \alpha_j \geq 0, j = 1, \ldots, c - 1\}$. Obviously this model realizes the true marginal distribution $p(x)$, when

$$\alpha_k = p(y = k) := \alpha_k^*, \ \ k = 1, \ldots, c - 1.$$

When the Fisher score is derived at the true parameter, it achieves $D(\boldsymbol{f}) = 0$.

**Lemma 1.** *The Fisher score $\boldsymbol{f}(x) = \nabla_{\boldsymbol{\alpha}} \log q_0(x|\boldsymbol{\alpha}^*)$ achieves $D(\boldsymbol{f}) = 0$.*

(proof) To prove the lemma, it is sufficient to show the existence of $(c-1) \times d$ matrix $W$ and $c-1$ dimensional vector $\boldsymbol{b}$ such that

$$W \nabla_{\boldsymbol{\alpha}} \log q_0(x|\boldsymbol{\alpha}^*) + \boldsymbol{b} = (p(y=1|x), \dots, p(y=c-1|x))^{\top}. \qquad (2.3)$$

The Fisher score for $q_0(x|\boldsymbol{\alpha}^*)$ is

$$\frac{\partial \log q_0(x|\boldsymbol{\alpha}^*)}{\partial \alpha_k} = \frac{p(y=k|x)}{\alpha_k^*} - \frac{p(y=c|x)}{1 - \sum_{k=1}^{c-1} \alpha_k^*}, \quad k = 1, \dots, c-1.$$

Let $A = A_0 + \gamma \mathbf{1}_{c-1,c-1}$ where

$$A_0 = \operatorname{diag}(\frac{1}{\alpha_1^*}, \dots, \frac{1}{\alpha_{c-1}^*}), \qquad \gamma = \frac{1}{1 - \sum_{j=1}^{c-1} \alpha_j^*}$$

and $\mathbf{1}_{mn}$ denotes $m \times n$ matrix filled with ones. Then

$$\nabla_{\boldsymbol{\alpha}} \log q_0(x|\boldsymbol{\alpha}^*) = A(p(y=1|x), \dots, p(y=c-1|x))^{\top} - \gamma \mathbf{1}_{c-1,1}.$$

When we set $W = A^{-1}$ and $\boldsymbol{b} = \gamma A^{-1} \mathbf{1}_{c-1,1}$, (2.3) holds. □

**Loose Models and Nuisance Dimensions**   We assumed that $p(x)$ is known but still we do not know the true class distributions $p(x|y)$. Thus the model $q_0(x|\boldsymbol{\alpha})$ in Lemma 1 is never available. In the following, the result of Lemma 1 is relaxed to a more general class of probability models by means of the chain rule of derivatives. However, in this case, we have to pay the price: nuisance dimensions.

Denote by $\mathcal{M}$ a set of probability distributions $\mathcal{M} = \{q_0 \mid q_0(x|\boldsymbol{\alpha}), \boldsymbol{\alpha} \in \mathcal{A}\}$. According to the information geometry [1], $\mathcal{M}$ is regarded as a manifold in a Riemannian space. Let $\mathcal{Q}$ denote the manifold of $q(x|\boldsymbol{\theta})$: $\mathcal{Q} = \{q \mid q(x|\boldsymbol{\theta}), \boldsymbol{\theta} \in \Re^d\}$. Now the question is how to determine a manifold $\mathcal{Q}$ such that $D(\boldsymbol{f}) = 0$, which is answered by the following theorem.

**Theorem 1.** *Assume that the true distribution $p(x)$ is contained in $\mathcal{Q}$:*

$$p(x) = q(x|\boldsymbol{\theta}^*) = q_0(x|\boldsymbol{\alpha}^*), \quad x \in \mathcal{X},$$

*where $\boldsymbol{\theta}^*$ is the true parameter. If the tangent space of $\mathcal{Q}$ at $p(x)$ contains the tangent space of $\mathcal{M}$ at the same point (Fig. 1), then the Fisher score $\boldsymbol{f}$ derived from $q(x|\boldsymbol{\theta}^*)$ satisfies $D(\boldsymbol{f}) = 0$.*

(proof) To prove the theorem, it is sufficient to show the existence of $(c-1) \times d$ matrix $W$ and $c-1$ dimensional vector $\boldsymbol{b}$ such that

$$W \nabla_{\boldsymbol{\theta}} \log q(x|\boldsymbol{\theta}) + \boldsymbol{b} = (p(y=1|x), \dots, p(y=c-1|x))^{\top}. \qquad (2.4)$$

When the tangent space of $\mathcal{M}$ is contained in that of $\mathcal{Q}$ around $p(x)$, we have the following by the chain rule:

$$\frac{\partial \log q_0(x|\boldsymbol{\alpha}^*)}{\partial \alpha_k} = \sum_{j=1}^{d} \frac{\partial \log q(x|\boldsymbol{\theta}^*)}{\partial \theta_j} \left. \frac{\partial \theta_j}{\partial \alpha_k} \right|_{\alpha_k = \alpha_k^*}. \qquad (2.5)$$

Let $U := [u_{ij}]$ where $u_{ij} = \left. \frac{\partial \theta_j}{\partial \alpha_i} \right|_{\alpha_i = \alpha_i^*}$. With this notation, (2.5) is rewritten as

$$U \nabla_{\boldsymbol{\theta}} \log q(x|\boldsymbol{\theta}^*) = A(p(y=1|x), \dots, p(y=c-1|x))^{\top} - \gamma \mathbf{1}_{c-1,1}$$

The equation (2.4) holds by setting $W = A^{-1}U$ and $\boldsymbol{b} = \gamma A^{-1} \mathbf{1}_{c-1,1}$. □

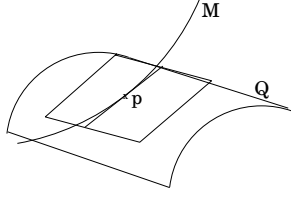

Figure 1: Information geometric picture of a probabilistic model whose Fisher score can fully extract the class information. When the tangent space of $\mathcal{M}$ is contained in $\mathcal{Q}$, the Fisher score can fully extract the class information, i.e. $D(\boldsymbol{f}) = 0$. Details explained in the text.

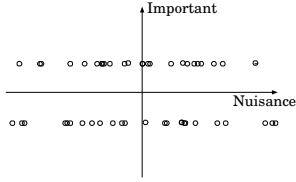

Figure 2: Feature space constructed by the Fisher score from the samples with two distinct clusters. The $x$ and $y$-axis corresponds to an nuisance and an important dimension, respectively. When the Euclidean metric is used as in K-Means, it is difficult to recover the two "lines" as clusters.

In determination of $q(x|\boldsymbol{\theta})$, we face the following dilemma: For capturing important dimensions (i.e. the tangent space of $\mathcal{M}$), the number of parameters $d$ should be sufficiently larger than $c$. But a large $d$ leads to a lot of nuisance dimensions, which are harmful for clustering in the feature space. In typical supervised classification experiments with hidden markov models [5, 15, 14], the number of parameters is much larger than the number of classes. However, in supervised scenarios, the existence of nuisance dimensions is not a serious problem, because advanced supervised classifiers such as the support vector machine have a built-in feature selector [7]. However in unsupervised scenarios without class labels, it is much more difficult to ignore nuisance dimensions. Fig. 2 shows how the feature space looks like, when the number of clusters is two and only one nuisance dimension is involved. Projected on the important dimension, clusters will be concentrated into two distinct points. However, when the Euclidean distance is adopted as in K-Means, it is difficult to recover true clusters because two "lines" are close to each other.

## 3   Clustering Algorithm for the Fisher score

In this section, we will develop a new clustering algorithm for the Fisher score. Let $\{y_i\}_{i=1}^{n} \in \mathcal{Y}$ be a set of class labels assigned to $\{x_i\}_{i=1}^{n} \in \mathcal{X}$, respectively. The purpose of clustering is to obtain $\{y_i\}_{i=1}^{n}$ only from samples $\{x_i\}_{i=1}^{n}$. As mensioned before, in clustering with the Fisher score, it is necessary to capture important dimensions. So far, it has been implemented as projection pursuit methods [3], which use general measures for interestingness, e.g. nongaussianity. However, from the last section's analysis, we know more than nongaussianity about important dimensions of the Fisher score. Thus we will construct a method specially tuned for the Fisher score.

Let us assume that the underlying classes are well separated, i.e. $p(y = k|x_i)$ is close to 0 or 1 for each sample $x_i \in \mathcal{X}$. When the class information is fully preserved, i.e. $D(\boldsymbol{f}) = 0$, there are $c$ bases in the space of the Fisher score, such that the samples in the $k$-th cluster are projected close to 1 on the $k$-th basis and the others are projected close to 0. The objective function of our clustering algorithm is designed to detect such bases:

$$\min_{y_1,\ldots,y_n} \min_{\boldsymbol{w}_1,\ldots,\boldsymbol{w}_c} \min_{b_1,\ldots,b_c} \sum_{k=1}^{c} \sum_{i=1}^{n} \left( \boldsymbol{w}_k^\top \boldsymbol{f}(x_i) + b_k - \delta(y_i = k) \right)^2, \qquad (3.1)$$

where $\delta(t)$ is the indicator function which is 1 if the condition $t$ holds and 0 otherwise. Notice that the optimal result of (3.1) is invariant to any invertible linear transformation $\boldsymbol{f}'(x) = A\boldsymbol{f}(x) + \boldsymbol{b}$. In contrast, K-means type methods are quite sensitive to linear transformation or data normalization [6]. When linear transformation is notoriously set,

K-means can end up with a false result which may not reflect the underlying structure.[1]

The objective function (3.1) can be minimized by the following EM-like alternating procedure:

1. Initialization: Set $\{y_i\}_{i=1}^n$ to initial values. Compute $\boldsymbol{\mu} = \frac{1}{n}\sum_{i=1}^n \boldsymbol{f}(x)$ and $C^{-1} = \left[\frac{1}{n}\sum_{i=1}^n \boldsymbol{f}(x_i)\boldsymbol{f}(x_i)^\top - \boldsymbol{\mu}\boldsymbol{\mu}^\top\right]^{-1}$ for later use.

2. Repeat 3. and 4. until the convergence of $\{y_i\}_{i=1}^n$.

3. Fix $\{y_i\}_{i=1}^n$ and minimize with respect to $\{\boldsymbol{w}_k\}_{k=1}^c$ and $\{b_k\}_{k=1}^c$. Each $\boldsymbol{w}_k, b_k$ is obtained as the solution of the following problem:

$$[\boldsymbol{w}_k, b_k] = \mathrm{argmin}_{\boldsymbol{w},b} \sum_{i=1}^n \left(\boldsymbol{w}^\top \boldsymbol{f}(x_i) + b - \delta(y_i = k)\right)^2.$$

This problem is analytically solved as

$$\boldsymbol{w}_k = C^{-1}(\frac{1}{n}\sum_{i=1}^n \delta(y_i = k)\boldsymbol{f}(x_i) - r_k\boldsymbol{\mu}), \quad b_k = r_k - \frac{1}{n}\sum_{i=1}^n (\boldsymbol{w}_k)^\top \boldsymbol{f}(x_i),$$

where $r_k = \frac{1}{n}\sum_{i=1}^n \delta(y_i = k)$.

4. Fix $\{\boldsymbol{w}_k\}_{k=1}^c, \{b_k\}_{k=1}^c$ and minimize with respect to $\{y_i\}_{i=1}^n$. Each $y_i$ is obtained by solving the following problem

$$y_i = \min_y \sum_{k=1}^c \left(\boldsymbol{w}_k^\top \boldsymbol{f}(x_i) + b_k - \delta(y = k)\right)^2$$

The solution can be obtained by exhaustive search.

Steps 1, 3, 4 take $O(nd^3)$, $O(ncd^2)$, $O(nc^2d)$ computational costs, respectively. Since the computational cost of algorithm is linear in $n$, it can be applied to problems with large sample sizes. This algorithm requires $O(d^3)$ time for inverting the matrix $C$, which may only be an obstacle for an application in an extremely high dimensional data setting.

## 4 Clustering Artificial Data

We will perform a clustering experiment with artificially generated data (Fig. 3). Since this data has a complicated structure, the Gaussian mixture with $m = 8$ components is used as a probabilistic model for the Fisher score: $q(x|\boldsymbol{\theta}) = \sum_{j=1}^m \gamma_j g(x|\boldsymbol{\mu}_j, C_j)$, where $g(x|\boldsymbol{\mu}, C)$ denotes the Gaussian distribution with mean $\boldsymbol{\mu}$ and covariance matrix $C$. The parameters are learned with the EM algorithm and the marginal distribution is accurately estimated as shown in Fig. 3 (upperleft). We applied the proposed algorithm and K-Means to the Fisher score calculated by taking derivatives with respect to $\gamma_j$. In order to have an initial partition, we first divided the points into 8 subclusters by the posterior probability to each Gaussian. In K-means and our approach defined in Sec. 3, initial clusters are constructed by randomly combining these subclusters. For each method, we chose the best result which achieved the minimum loss among the local minima obtained from 100 clustering experiments. As a result, the proposed method obtained clearly separated clusters (Fig. 3, upper right) but K-Means failed to recover the "correct" clusters, which is considered as the effect of nuisance dimensions (Fig. 3, lower left). When the Fisher score is *whitened* (i.e. linear transformation to have mean 0 and unit covariance matrix), the result of K-Means changed to Fig. 3 (lowerright) but the solution of our method stayed the same as discussed in Sec. 3. Of course, this kind of problem can be solved by many state-of-the-art methods (e.g. [9, 8])

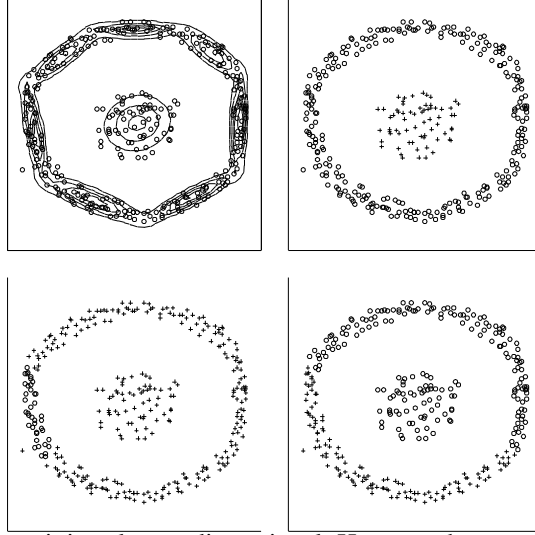

Figure 3: (Upperleft) Toy dataset used for clustering. Contours show the estimated density with the mixture of 8 Gaussians. (Upperright) Clustering result of the proposed algorithm. (Lowerleft) Result of K-Means with the Fisher score. (Lowerright) Result of K-Means with the whitened Fisher score.

because it is only two dimensional. However these methods typically do not scale to large dimensional or discrete problems. Standard mixture modeling methods have difficulties in modeling such complicated cluster shapes [9, 10]. One straightforward way is to model each cluster as a Gaussian Mixture: $p(x) = \sum_{k=1}^{c} \alpha_k \sum_{\ell=1}^{m_k} \beta_{k\ell} g(x|\boldsymbol{\mu}_{k\ell}, C_{k\ell})$. However, special care needs to be taken for such a "mixture of mixtures" problem. When the parameters $\alpha_k, \beta_{k\ell}, \boldsymbol{\mu}_{kl}$ and $C_{kl}$ are jointly optimized in a maximum likelihood process, the solution is not unique. In order to have meaningful results e.g. in our dataset, one has to constrain the parameters such that 8 Gaussians form 2 groups. In the Bayesian framework, this can be done by specifying an appropriate prior distributions on parameters, which can become rather involved. Roberts et. al. [10] tackled this problem by means of the minimum entropy principle using MCMC which is somewhat more complicated than our approach.

## 5 Clustering Amino Acid Sequences

In this section, we will apply our method to cluster bacterial *gyrB* amino acid sequences, where the hidden markov model (HMM) is used to derive the Fisher score. *gyrB* - gyrase subunit B - is a DNA topoisomerase (type II) which plays essential roles in fundamental mechanisms of living organisms such as DNA replication, transcription, recombination and repair etc. One more important feature of *gyrB* is its capability of being an evolutionary and taxonomic marker alternating popular 16S rRNA [17]. Our data set consists of 55 amino acid sequences containing three clusters (9,32,14). The three clusters correspond to three genera of high GC-content gram-positive bacteria, that is, Corynebacteria, Mycobacteria, Rhodococcus, respectively. Each sequence is represented as a sequence of 20 characters, each of which represents an amino acid. The length of each sequence is different from 408 to 442, which makes it difficult to convert a sequence into a vector of fixed dimensionality.

In order to evaluate the partitions we use the Adjusted Rand Index (ARI) [4, 18]. Let $U_1, \ldots, U_c$ be the obtained clusters and $T_1, \ldots, T_s$ be the ground truth clusters. Let $n_{ij}$ be the number of samples which belongs to both $U_i$ and $T_j$. Also let $n_{i\cdot}$ and $n_{\cdot j}$ be the number of samples in $U_i$ and $T_j$, respectively. ARI is defined as

$$\frac{\sum_{i,j} \binom{n_{ij}}{2} - \left[\sum_i \binom{n_{i\cdot}}{2} \sum_j (n_{\cdot j}2)\right] / \binom{n}{2}}{\frac{1}{2}\left[\sum_i \binom{n_{i\cdot}}{2} + \sum_j \binom{n_{\cdot j}}{2}\right] - \left[\sum_i \binom{n_{i\cdot}}{2} \sum_j \binom{n_{\cdot j}}{2}\right] / \binom{n}{2}}$$

The attractive point of ARI is that it can measure the difference of two partitions even when

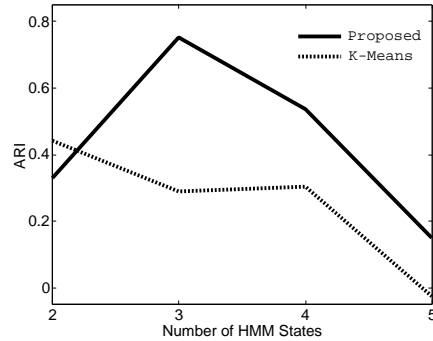

Figure 4: Adjusted Rand indices of K-Means and the proposed method in a sequence classification experiment.

the number of clusters is different. When the two partitions are exactly the same, ARI is 1, and the expected value of ARI over random partitions is 0 (see [4] for details).

In order to derive the Fisher score, we trained complete-connection HMMs via the Baum-Welch algorithm, where the number of states $s$ is changed from 2 to 5, and each state emits one of $t = 20$ characters. This HMM has $s$ initial state probabilities, $s$ terminal state probabilities, $s^2$ transition probabilities and $st$ emission probabilities. Thus when $s = 3$ for example, a HMM has 75 parameters in total, which is much larger than the number of potential classes (i.e. 3). The derivative is taken with respect to all paramaters as described in detail in [15]. Notice that we did not perform any normalization to the Fisher score vectors. In order to avoid local minima, we tried 1000 different initial values and chose the one which achieved the minimum loss both in K-means and our method. In K-Means, initial centers are sampled from the uniform distribution in the smallest hypercube which contains all samples. In the proposed method, every $w_{ki}$ is sampled from the normal distribution with mean 0 and standard deviation 0.001. Every $b_k$ is initially set to zero.

Fig. 4 shows the ARIs of two methods against the number of HMM states. Our method shows the highest ARI (0.754) when the number of HMM states is 3, which shows that important dimensions are successfully discovered from the "sea" of nuisance dimensions. In contrast, the ARI of K-Means decreases monotonically as the number of HMM states increases, which shows the K-Means is not robust against nuisance dimensions. But when the number of nuisance dimensions are too many (i.e. $s = 4, 5$), our method was caught in false clusters which happened to appear in nuisance dimensions. This result suggests that prior dimensionality reduction may be effective (cf.[11]), but it is beyond the scope of this paper.

## 6  Concluding Remarks

In this paper, we illustrated how the class information is encoded in the Fisher score: most information is packed in a few dimensions and there are a lot of nuisance dimensions. Advanced supervised classifiers such as the support vector machine have a built-in feature selector [7], so they can detect important dimensions automatically. However in unsupervised learning, it is not easy to detect important dimensions because of the lack of class labels. We proposed a novel very simple clustering algorithm that can ignore nuisance dimensions and tested it in artificial and real data experiments. An interesting aspect of our gyrB experiment is that the ideal scenario assumed in the theory section is not fulfilled anymore as clusters might overlap. Nevertheless our algorithm is robust in this respect and achieves highly promising results.

The Fisher score derives features using the prior knowledge of the marginal distribution. In general, it is impossible to infer anything about the conditional distribution $p(y|x)$ from the marginal $p(x)$ [12] without any further assumptions. However, when one knows the directions that the marginal distribution can move (i.e. the model of marginal distribution), it is possible to extract information about $p(y|x)$, even though it may be corrupted by many nuisance dimensions. Our method is straightforwardly applicable to the objects to which the Fisher kernel has been applied (e.g. speech signals [13] and documents [16]).

**Acknowledgement**   The authors gratefully acknowledge that the bacterial *gyrB* amino acid sequences are offered by courtesy of Identification and Classification of Bacteria (ICB) database team [17]. KRM thanks for partial support by DFG grant # MU 987/1-1.

## Footnotes

[1]When the covariance matrix of each cluster is allowed to be different in K-Means, it becomes invariant to normalization. However this method in turn causes singularities, where a cluster shrinks to the delta distribution, and difficult to learn in high dimensional spaces.

# References

[1] S. Amari and H. Nagaoka. *Methods of Information Geometry*, volume 191 of *Translations of Mathematical Monographs*. American Mathematical Society, 2001.

[2] R. Durbin, S. Eddy, A. Krogh, and G. Mitchison. *Biological Sequence Analysis: Probabilistic Models of Proteins and Nucleic Acids*. Cambridge University Press, 1998.

[3] P.J. Huber. Projection pursuit. *Annals of Statistics*, 13:435–475, 1985.

[4] L. Hubert and P. Arabie. Comparing partitions. *J. Classif.*, pages 193–218, 1985.

[5] T.S. Jaakkola and D. Haussler. Exploiting generative models in discriminative classifiers. In M.S. Kearns, S.A. Solla, and D.A. Cohn, editors, *Advances in Neural Information Processing Systems 11*, pages 487–493. MIT Press, 1999.

[6] A.K. Jain and R.C. Dubes. *Algorithms for Clustering Data*. Prentice Hall, 1988.

[7] K.-R. Müller, S. Mika, G. Rätsch, K. Tsuda, and B. Schölkopf. An introduction to kernel-based learning algorithms. *IEEE Trans. Neural Networks*, 12(2):181–201, 2001.

[8] A. Y. Ng, M. I. Jordan, and Y. Weiss. On spectral clustering: Analysis and an algorithm. In T. G. Dietterich, S. Becker, and Z. Ghahramani, editors, *Advances in Neural Information Processing Systems 14*. MIT Press, 2002.

[9] M. Rattray. A model-based distance for clustering. In *Proc. IJCNN'00*, 2000.

[10] S.J. Roberts, C. Holmes, and D. Denison. Minimum entropy data partitioning using reversible jump markov chain monte carlo. *IEEE Trans. Patt. Anal. Mach. Intell.*, 23(8):909–915, 2001.

[11] V. Roth, J. Laub, J.M. Buhmann, and K.-R. Müller. Going metric: Denoising pairwise data. In *NIPS02*, 2003. to appear.

[12] M. Seeger. Learning with labeled and unlabeled data. Technical report, Institute for Adaptive and Neural Computation, University of Edinburgh, 2001. http://www.dai.ed.ac.uk/homes/seeger/papers/review.ps.gz.

[13] N. Smith and M. Gales. Speech recognition using SVMs. In T.G. Dietterich, S. Becker, and Z. Ghahramani, editors, *Advances in Neural Information Processing Systems 14*. MIT Press, 2002.

[14] S. Sonnenburg, G. Rätsch, A. Jagota, and K.-R. Müller. New methods for splice site recognition. In *ICANN'02*, pages 329–336, 2002.

[15] K. Tsuda, M. Kawanabe, G. Rätsch, S. Sonnenburg, and K.-R. Müller. A new discriminative kernel from probabilistic models. *Neural Computation*, 14(10):2397–2414, 2002.

[16] A. Vinokourov and M. Girolami. A probabilistic framework for the hierarchic organization and classification of document collections. *Journal of Intelligent Information Systems*, 18(2/3):153–172, 2002.

[17] K. Watanabe, J.S. Nelson, S. Harayama, and H. Kasai. ICB database: the gyrB database for identification and classification of bacteria. *Nucleic Acids Res.*, 29:344–345, 2001.

[18] K.Y. Yeung and W.L. Ruzzo. Principal component analysis for clustering gene expression data. *Bioinformatics*, 17(9):763–774, 2001.
